# Dual Mechanisms for Neural Binding and Segmentation

**Paul Sajda and Leif H. Finkel**
Department of Bioengineering and
Institute of Neurological Science
University of Pennsylvania
220 South 33rd Street
Philadelphia, PA. 19104-6392

## Abstract

We propose that the binding and segmentation of visual features is mediated by two complementary mechanisms; a low resolution, spatial-based, resource-free process and a high resolution, temporal-based, resource-limited process. In the visual cortex, the former depends upon the orderly topographic organization in striate and extrastriate areas while the latter may be related to observed temporal relationships between neuronal activities. Computer simulations illustrate the role the two mechanisms play in figure/ground discrimination, depth-from-occlusion, and the vividness of perceptual completion.

## 1  COMPLEMENTARY BINDING MECHANISMS

The "binding problem" is a classic problem in computational neuroscience which considers how neuronal activities are grouped to create mental representations. For the case of visual processing, the binding of neuronal activities requires a mechanism for selectively grouping fragmented visual features in order to construct the coherent representations (i.e. objects) which we perceive. In this paper we argue for the existence of two complementary mechanisms for neural binding, and we show how such mechanisms may operate in the construction of intermediate-level visual representations.

Ordered cortical topography has been found in both striate and extrastriate areas and is believed to be a fundamental organizational principle of visual cortex. One functional role for this topographic mapping may be to facilitate a spatial-based binding system. For example, active neurons or neural populations within a cortical area could be grouped together based on topographic proximity while those in different areas could be grouped if they lie in rough topographic register. An advantage of this scheme is that it can be carried out in parallel across the visual field. However, a spatial-based mechanism will tend to bind overlapping or occluded objects which should otherwise be segmented. An alternative binding mechanism is therefore necessary for binding and segmenting overlapping objects and surfaces.

Temporal binding is a second type of neural binding. Temporal binding differs from spatial binding in two essential ways; 1) it operates at high spatial resolutions and 2) it binds and segments in the temporal domain, allowing for the coexistence of multiple objects in the same topographic region. Others have proposed that temporal events, such as phase-locked firing or $\gamma$ oscillations may play a role in neural binding (von der Malsburg, 1981; Gray and Singer, 1989, Crick and Koch, 1990). For purposes of this discussion, we do not consider the specific nature of the temporal events underlying neural binding, only that the binding itself is temporally dependent. The disadvantage of operating in the temporal domain is that the biophysical properties of cortical neurons (e.g. membrane time constants) forces this processing to be resource-limited–only a small number of objects or surfaces can be bound and segmented simultaneously.

## 2   COMPUTING INTERMEDIATE-LEVEL VISUAL REPRESENTATIONS: DIRECTION OF FIGURE

We consider how these two classes of binding can be used to compute context-dependent (non-local) characteristics about the visual scene. An example of a context-dependent scene characteristic is contour ownership or *direction of figure*. Direction of figure is a useful intermediate-level visual representation since it can be used to organize an image into a perceptual scene (e.g. infer relative depth and link segregated features). Figure 1A illustrates the relationship between contours and surfaces implied by direction of figure.

We describe a model which utilizes both spatial and temporal binding to compute direction of figure (DOF). Prior to computing the DOF, the surface contours in the image are extracted. These contours are then temporally bound by a process we call "contour binding" (Finkel and Sajda, 1992). In the model, the temporal properties of the units are represented by a *temporal binding value*. We will not consider the details of this process except to say that units with similar temporal binding values are bound together while those with different values are segmented. *In vivo*, this temporal binding value may be represented by phase of neural firing, oscillation frequency, or some other specific temporal property of neuronal activity.

The DOF is computed by circuitry which is organized in a columnar structure, shown in figure 2A. There are two primary circuits which operate to compute the direction of figure; one being a temporal-dependent/spatial-independent (TDSI) circuit selective to "closure", the other a spatial-dependent/temporal-independent

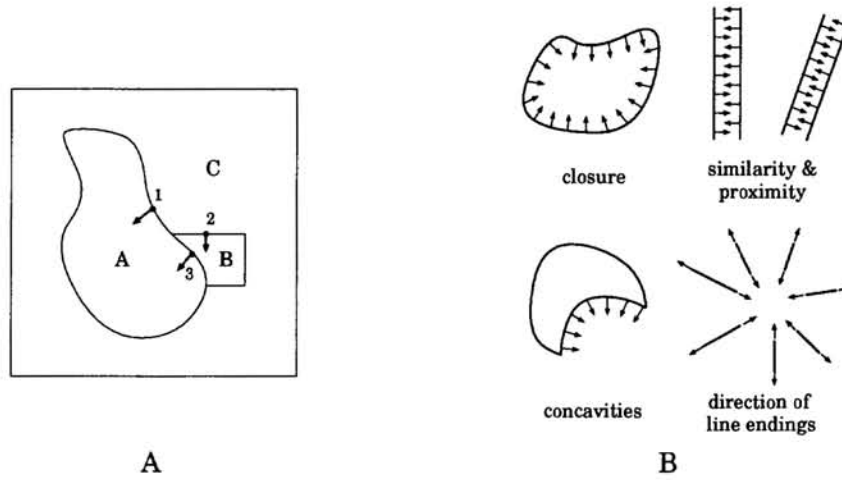

closure

similarity & proximity

concavities

direction of line endings

A

B

Figure 1: **A** Direction of figure as a surface representation. At point (1) the contour belongs to the surface contour of region $A$ and therefore $A$ owns the contour. This relationship is represented locally as a "direction of figure" vector pointing toward region $A$. Additional ownership relationships are shown for points (2) and (3). **B** Cues used in determining direction of figure.

(SDTI) circuit selective to "similarity and proximity". There are also two secondary circuits which play a transient role in determining direction of figure. One is based on the observation that concave segments bounded by discontinuities are a cue for occlusion and ownership, while the other considers the direction of line endings as a potential cue. Figure 1B summarizes the cues used to determine direction of figure. In this paper, we focus on the TDSI and SDTI circuits since they best illustrate the nature of the dual binding mechanisms. The perceptual consequences of attributing closure discrimination to temporal binding and similarity/proximity to spatial binding is illustrated in figure 3.

## 2.1   TDSI CIRCUIT

Figure 2B(i) shows the neural architecture of the TDSI mechanism. The activity of the TDSI circuit selective for a direction of figure $\alpha$ is computed by comparing the amount of closure on either side of a contour. Closure is computed by summing the temporal dependent inputs over all directions $i$;

$$TDSI^\alpha = \left[ \sum_i s_i^\alpha(t_i) - \sum_i s_i^{\alpha-180^\circ}(t_i) \right]_0^1 \tag{1}$$

The brackets ([ ]) indicate an implicit thresholding (if $x < 0$ then $[x] = 0$, otherwise $[x] = x$) and $s_i^\alpha(t_i)$ and $s_i^{\alpha-180^\circ}(t_i)$ are the temporal dependent inputs, computed as;

$$s_i^\alpha(t_i) = \begin{cases} 1 & \text{if} \begin{cases} (s_j > S_T) \\ \text{and} \\ ((t_i - \Delta t) < t_j < (t_i + \Delta t)) \end{cases} \\ 0 & \text{otherwise} \end{cases} \tag{2}$$

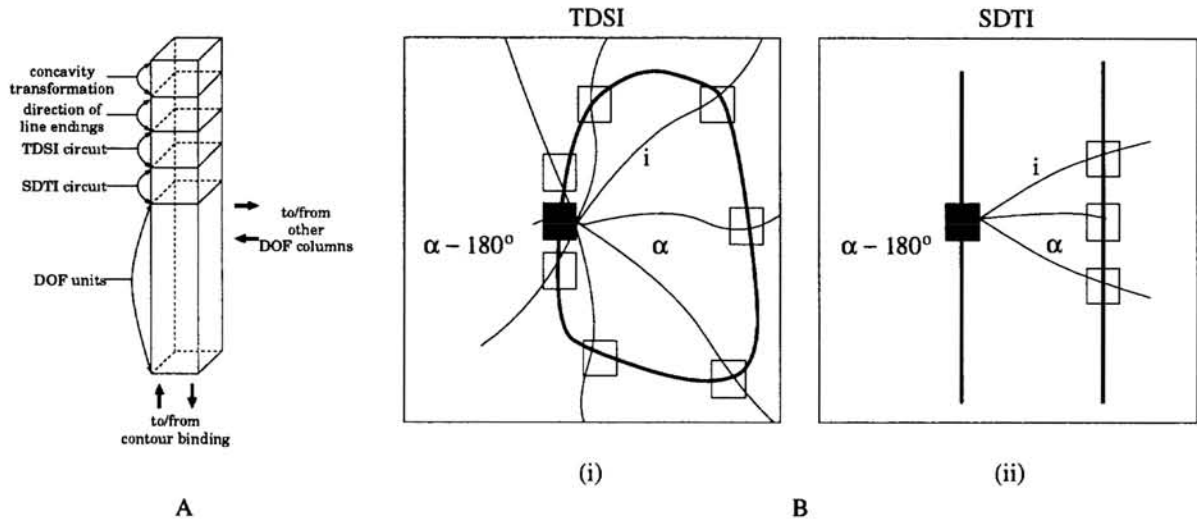

Figure 2: **A** Divisions, inputs, and outputs for a DOF column. **B** The two primary circuits operating to compute direction of figure. *(i)* Top view of temporal-dependent/spatial-independent (TDSI) circuit architecture. Filled square represents position of a specific column in the network. Unfilled squares represent other DOF columns serving as input to this column. Bold curve corresponds to a surface contour in the input. Shown is the pattern of long-range horizontal connections converging on the right side of the column (side $\alpha$). *(ii)* Top view of spatial-dependent/temporal-independent (SDTI) circuit architecture. Shown is the pattern of connections converging on the right side of the column (side $\alpha$).

where $\alpha$ and $\alpha - 180°$ represent the regions on either side of the contour, $s_j$ is the activation of a unit along the direction $i$ (For simulations $i$ varies between $0°$ and $315°$ by increments of $45°$), $\Delta t$ determines the range of temporal binding values over which the column will integrate input, and $S_T$ is the activation threshold. The temporal dependence of this circuit implies that only those DOF columns having the same temporal binding value affect the closure computation.

## 2.2  SDTI CIRCUIT

Figure 2B(ii) illustrates the neural architecture of the SDTI mechanism. The SDTI circuit organizes elements in the scene based on "proximity" and "similarity" of orientation. Unlike the TDSI circuit which depends upon temporal binding, the SDTI circuit uses spatial binding to access information across the network.

Activity is integrated from units with similar orientation tuning which lie in a direction orthogonal to the contour (i.e. from parallel line segments). The activity of the SDTI circuit selective for a direction of figure $\alpha$ is computed by comparing input from similar orientations on either side of a contour;

$$SDTI^\alpha = \frac{1}{S_{max}} \left( \sum_i s_i^\alpha(\theta_i) - \sum_i s_i^{\alpha-180°}(\theta_i) \right) \qquad (3)$$

where $S_{max}$ is a constant for normalizing the SDTI activity between 0 and 1 and $s_i^\alpha(\theta_i)$ and $s_i^{\alpha-180°}(\theta_i)$ are spatial dependent inputs selective for an orientation $\theta$,

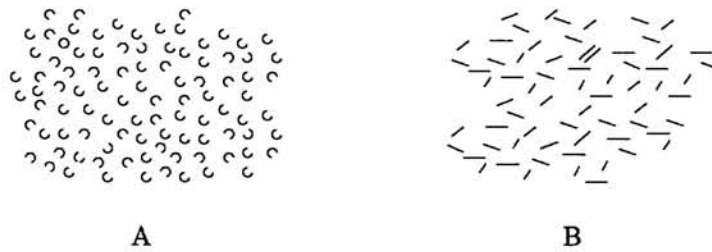

Figure 3: **A** The model predicts that a closed figure could not be discriminated in parallel search since its detection depends on resource-limited temporal binding. **B** Conversely, proximal parallel segments are predicted to be discriminated in parallel search due to resource-free spatial binding.

computed as;

$$
s_i^\alpha(\theta_i) = \begin{cases} c_{ij} s_j(x, y, \theta_j) & \text{if} \begin{cases} (s_j(x, y, \theta_j) > S_T) \\ \text{and} \\ (\theta_i = \theta_j) \end{cases} \\ 0 & \text{otherwise} \end{cases} \tag{4}
$$

where $\alpha$ and $\alpha - 180^o$ represent the regions on either side of the contour, $\theta$ is the orientation of the contour, $i$ is the direction from which the unit receives input, $c_{ij}$ is the connection strength ($c_{ij}$ falls off as a gaussian with distance), and $s_j(x, y, \theta_j)$ is the activation of a unit along the direction $i$ which is mapped to retinotopic location $(x, y)$ and selective for an orientation $\theta_j$ (For simulations $i$ varies between the following three angles; $\perp \theta_i, \perp (\theta_i - 45^o), \perp (\theta_i + 45^o)$). Since the efficacy of the connections, $c_{ij}$, decrease with distance, columns which are further apart are less likely to be bound together. Neighboring parallel contours generate the greatest activation and the circuit tends to discriminate the region between the two parallel contours as the figure.

## 2.3   COMPUTED DOF

The activity of a direction of figure unit representing a direction $\alpha$ is given by the sum of the four components;

$$
DOF^\alpha = C_1(TDSI^\alpha) + C_2(SDTI^\alpha) + C_3(CON^\alpha) + C_4(DLE^\alpha) \tag{5}
$$

where the constants define the contribution of each cue to the computed DOF. Note that in this paper we have not considered the mechanisms for computing the DOF given the two secondary cues (concavities ($CON^\alpha$) and direction of line endings ($DLE^\alpha$)). The DOF activation is computed for all directions $\alpha$ (For simulations $\alpha$ varies between $0^o$ and $315^o$ by increments of $45^o$) with the direction producing the largest activation representing the direction of figure.

## 3   SIMULATION RESULTS

The following are simulations illustrating the role the dual binding mechanisms play in perceptual organization. All simulations were carried out using the NEXUS Neural Simulation Environment (Sajda and Finkel, 1992).

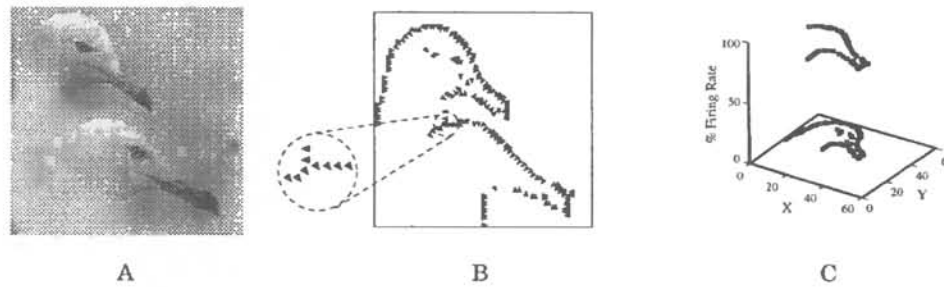

Figure 4: **A** 128x128 pixel grayscale image. **B** Direction of figure computed by the network. Direction of figure is shown as an oriented arrowhead, where the orientation represents the preferred direction of the DOF unit which is most active. **C** Depth of surfaces. Direction of figure relationships (such as those in the inset of *B*) are used to infer relative depth. Plot shows % activity of units in the foreground network–higher activity implies that the surface is closer to the viewer.

## 3.1   FIGURE/GROUND AND DEPTH-FROM-OCCLUSION

Figure 4A is a grayscale image used as input to the network. Figure 4B shows the direction of figure computed by the model. Note that though the surface contours are incomplete, the model is still able to characterize the direction of figure and distinguish figure/ground over most of the contour. This is in contrast to models proposing diffusion-like mechanisms for determining figure/ground relationships which tend to fail if complete contour closure is not realized.

The model utilizes direction of figure to determine occlusion relationships and stratify objects in relative depth, results shown in figure 4C. This method of inferring the relative depth of surfaces given occlusion is in contrast to traditional approaches utilizing T-junctions. The obvious advantage of using direction of figure is that it is a context-dependent feature directly linked to the representation of surfaces.

## 3.2   VIVIDNESS OF PERCEPTUAL COMPLETION

Our previous work (Finkel and Sajda, 1992) has shown that direction of figure is important for completion phenomena, such as the construction of illusory contours and surfaces. More interestingly, our model offers an explanation for differences in perceived vividness between different inducing stimuli. For example, subjects tend to rank the vividness of the illusory figures in figure 5 from left to right, with the figure on the left being the most vivid and that on the right the least.

Our model accounts for this effect in terms of the magnitude of the direction of figure along the illusory contour. Figure 6 shows the individual components contributing to the direction of figure. For a typical inducer, such as the pacman in figure 6, the TDSI and SDTI circuits tend to force the direction of figure of the L-shaped segment to region 1 while the concavity/convexity transformation tries to force the direction of figure of the segment to be toward region 2. This transformation transiently overwhelms the TDSI and SDTI responses, so that the direction of figure of the L-shaped segment is toward region 2. However, the TDSI and SDTI activation will affect the magnitude of the direction of figure, as shown in figure 7. For example,

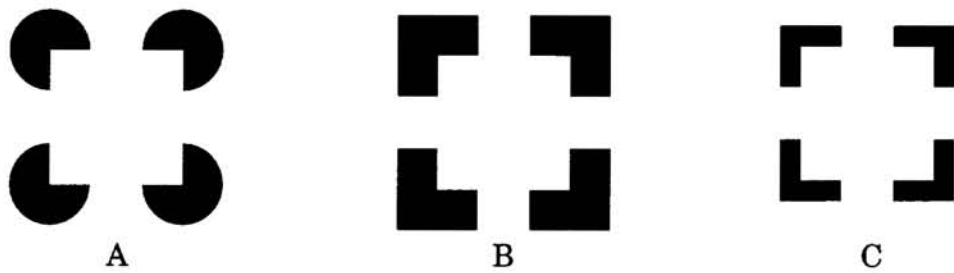

Figure 5: Illusory contour vividness as a function of inducer shape. Three types of inducers are arranged to generate an illusory square. **A** pacman inducer, **B** thick L inducer and **C** thin L inducer. Subjects rank the vividness of the illusory squares from left to right $((A) > (B) > (C))$.

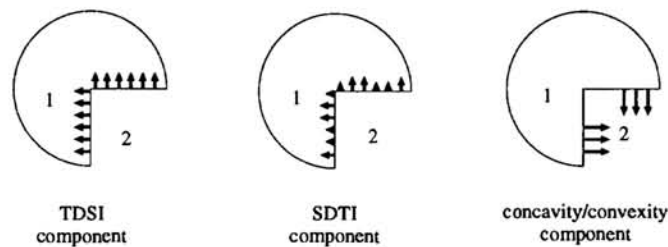

Figure 6: Processes contributing to the direction of figure of the L-shaped contour segment. The TDSI and SDTI circuits assign the contour to region 1, while the change of the concavity to a convexity assigns the segment to region 2.

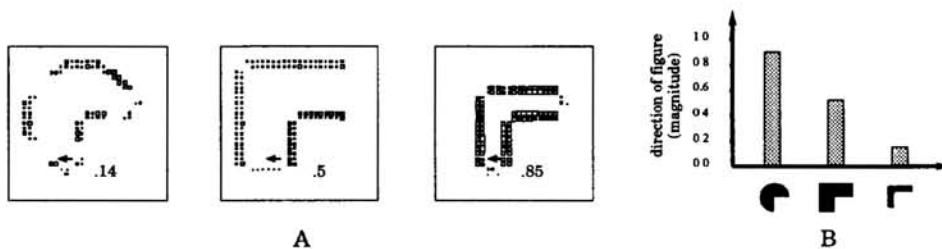

Figure 7: **A** Activity of SDTI units for the upper left inducer of each stimulus, where the area of each square is proportional to unit activity. The SDTI units try to assign the L-shaped segment to the region of the pacman. Numerical values indicates the magnitude of the SDTI effect. **B** Magnitude of direction of figure along the L-shaped segment as a function of inducer shape. The direction of figure in all cases is toward the region of the illusory square.

the weaker the activation of the TDSI and SDTI circuits, the stronger the activation of the DOF units assigning the L-shaped segment to region 2.

Referring back to the inducer types in figure 5, one can see that though the TDSI component is the same for all three inducers (i.e. all three generate the same amount of closure) the SDTI contribution differs, shown quantitatively in figure 7A. The contribution of the SDTI circuit is greatest for the thin L inducers and least for the pacmen inducers–the L-shaped segments for the pacman stimulus are more strongly owned by the surface of the illusory square than those for the thin L inducer. This is illustrated in figure 7B, a plot of the magnitude of the direction of figure for each inducer configuration. This result can be interpreted as the model's ordering of perceived vividness, which is consistent with that of human observers.

## 4   CONCLUSION

In this paper we have argued for the utility of binding neural activities in both the spatial and temporal domains. We have shown that a scheme consisting of these complementary mechanisms can be used to compute context-dependent scene characteristics, such as direction of figure. Finally, we have illustrated with computer simulations the role these dual binding mechanisms play in accounting for aspects of figure/ground perception, depth-from-occlusion, and perceptual vividness of illusory contours and surfaces. It is interesting to speculate on the relationship between these complementary binding mechanisms and the traditional distinction between preattentive and attentional perception.

**Acknowledgements**

This work is supported by grants from ONR (N00014-90-J-1864, N00014-93-1-0681), The Whitaker Foundation, and The McDonnell-Pew Program in Cognitive Neuroscience.

**References**

F. Crick and C. Koch. Towards a neurobiological theory of consciousness. *Seminars in Neuroscience*, 2:263–275, 1990.

L.H. Finkel and P. Sajda. Object discrimination based on depth-from-occlusion. *Neural Computation*, 4(6):901–921, 1992.

C. M. Gray and W. Singer. Neuronal oscillations in orientation columns of cat visual cortex. *Proceedings of the National Academy of Science USA*, 86:1698–1702, 1989.

P. Sajda and L. Finkel. NEXUS: A simulation environment for large-scale neural systems. *Simulation*, 59(6):358–364, 1992.

C. von der Malsburg. The correlation theory of brain function. Technical Report Internal Rep. No. 81-2, Max-Plank-Institute for Biophysical Chemistry, Department of Neurobiology, Gottingen, Germany, 1981.